# Grammar Learning by a Self-Organizing Network

**Michiro Negishi**
Dept. of Cognitive and Neural Systems, Boston University
111 Cummington Street
Boston, MA 02215 email : negishi@cns.bu.edu

## Abstract

This paper presents the design and simulation results of a self-organizing neural network which induces a grammar from example sentences. Input sentences are generated from a simple phrase structure grammar including number agreement, verb transitivity, and recursive noun phrase construction rules. The network induces a grammar explicitly in the form of symbol categorization rules and phrase structure rules.

## 1 Purpose and related works

The purpose of this research is to show that a self-organizing network with a certain structure can acquire syntactic knowledge from only positive (*i.e.* grammatical) data, without requiring any initial knowledge or external teachers that correct errors.

There has been research on supervised neural network models of language acquisition tasks [Elman, 1991, Miikkulainen and Dyer, 1988, John and McClelland, 1988]. Unlike these supervised models, the current model self-organizes word and phrasal categories and phrase construction rules through mere exposure to input sentences, without any artificially defined task goals. There also have been self-organizing models of language acquisition tasks [Ritter and Kohonen, 1990, Scholtes, 1991]. Compared to these models, the current model acquires phrase structure rules in more explicit forms, and it learns wider and more structured contexts, as will be explained below.

## 2 Network Structure and Algorithm

The design of the current network is motivated by the observation that humans have the ability to handle a frequently occurring sequence of symbols (chunk) as an unit of information [Grossberg, 1978, Mannes, 1993]. The network consists of two parts : classification networks and production networks (Figure 1). The classification networks categorize words and phrases, and the production networks

evaluate how it is likely for a pair of categories to form a phrase. A pair of combined categories is given its own symbol, and fed back to the classifiers.

After weights are formed, the network parses a sentence as follows. Input words are incrementally added to the neural sequence memory called the Gradient Field [Grossberg, 1978] (GF hereafter). The top (*i.e. most recent*) two symbols and the lookahead token are classified by three classification networks. Here a symbol is either a word or a phrase, and the lookahead token is the word which will be read in next. Then the lookahead token and the top symbol in the GF are sent to the right production network, and the top and the second ones are sent to the left production network. If the latter pair is judged to be more likely to form a phrase, the symbol pair *reduces* to a phrase, and the phrase is fed back to the GF after removing the top two symbols. Otherwise, the lookahead token is added to the sequence memory, causing a *shift* in the sequence memory. If the input sentence is grammatical, the repetition of this process reduces the whole sentence to a single "S" (sentence) symbol. The sequence of shifts and reductions (annoted with the resultant symbols) amounts to a parse of the sentence.

During learning, the operations stated above are carried out as weights are gradually formed. In classification networks, the weights record a distribution pattern with respect to each symbol. That is, the weights record the co-occurrence of up to three adjacent symbols in the corpus. An symbol is classified in terms of this distribution in the classification networks. The production networks keep track of the categories of adjacent symbols. If the occurrence of one category reliably predicts the next or the previous one, the pair of categories forms a phrase, and is given the status of an symbol which is treated just like a word in the sentence. Because the symbols include phrases, the learned context is wider and more structured than the mere bigram, as well as the contexts utilized in [Ritter and Kohonen, 1990, Scholtes, 1991].

# 3   Simulation

## 3.1   The Simulation Task

The grammar used to generate input sentences (Table 3) is identical to that used in [Elman, 1991], except that it does not include optionally transitive verbs and proper nouns. Lengths of the input sentences are limited to 16 words. To determine the completion of learning, after accepting 200 consecutive sentences with learning, learning is suppressed and other 200 sentences are processed to see if all are accepted. In addition, the network was tested for 44 ungrammatical sentences to see that they are correctly rejected. Ungrammatical sentences are derived by hand from randomly generated grammatical sentences. Parameters used in the simulation are : number of symbol nodes = 30 (words) + 250 (phrases), number of category nodes = 150, $\epsilon = 10^{-9}$, $\gamma = 0.25$, $\rho = 0.65$, $\alpha_1 = 0.00005$, $\beta_1 = 0.005$, $\beta_2 = 0.2$, $\alpha_3 = 0.0001$, $\beta_3 = 0.001$, and $T = 4.0$.

### 3.2 Acquired Syntax Rules

Learning was completed after learning 19800 grammatical sentences. Tables 1 and 2 show the acquired syntax rules extracted from the connection weights. Note that category names such as Ns, VPp, are not given a priori, but assigned by the author for the exposition. Only rules that eventually may reach the "S"(sentence) node are shown. There were a small number of uninterpretable rules, which are marked "?". These rules might disturb normal parsing for some sentences, but they were not activated while testing for 200 sentences after learning.

### 3.3 Discussion

Recursive noun phrase structures should be learned by finding equivalences of distribution between noun phrases and nouns. However, nouns and noun phrases have the same contextual features *only when* they are in certain contexts. An examination of the acquired grammar reveals that the network finds equivalence of features not of "N" and "N RC" (where RC is a relative clause) but of "N V" and "N RC V" (when "N RC" is subjective), or "V N" and "V N RC" (when "N RC" is objective). As an example, let us examine the parsing of the sentence [19912] below. The rule used to reduce *FEEDS CATS WHO LIVE* ("V N RC") is P0, which is classified as category C4, which includes P121 ("V N") where V are the singular forms of transitive verbs, and also includes the "V" where V are singular forms of intransitive verbs. Thus, *GIRL WHO FEEDS CATS WHO LIVE* is reduced to *GIRL WHO "VPsingle"*.

```
***[19912]*******************************************************
  +---141---+
  |     +---88------+
  |     |    +---206------+
  |     |    |     +----0----+
  |     |    |     |      +-219-+
  |     |  +-41-+  |    +-36-+  |
BOYS CHASE GIRL WHO FEEDS CATS WHO LIVE .

<<Accepted>> Top symbol was 77
```

## 4   Conclusion and Future Direction

In this paper, a self-organizing neural network model of grammar learning was presented. A basic principle of the network is that all words and phrases are categorized by the contexts in which they appear, and that familiar sequence of categories are chunked.

As it stands, the scope of the grammar used in the simulation is extremely limited. Also, considering the poverty of the actual learning environment, the learning of syntax should also be guided by the cognitive competence to comprehend the utterance situations and conversational contexts. However, being a self-organizing network, the current model offers a plausible model of natural language acquisition through mere exposures to only grammatical sentences, not requiring any external teacher or an explicit goal.

## Table 1. Acquired categorization rules

| | | | |
|---|---|---|---|
| S | := | C29 /* NPs VPs */ \| | |
| | | C30 /* ? */ \| | |
| | | C77 /* NPp VPp */ | |
| C4 | := | LIVES \| WALKS \| | |
| | | P0 /* VTs Np RC */ \| | |
| | | P74 /* VTs Ns RC */ \| | |
| | | P121 /* VTs Ns */ \| | |
| | | P157 /* VTs Np */ | = /* VPs */ |
| C13 | := | GIRL \| DOG \| | |
| | | CAT \| BOY | = /* Ns */ |
| C16 | := | CHASE \| FEED | = /* VTp */ |
| C18 | := | WHO | = /* R */ |
| C20 | := | CHASES \| FEEDS | = /* VTs */ |
| C26 | := | BOYS \| CATS \| | |
| | | DOGS \| GIRLS | = /* Np */ |
| C29 | := | P93 /* Ns RC VPs */ \| | |
| | | P138 /* Ns VPs */ | = /* NPs VPs */ |
| C30 | := | P2 /* VTp NPp VPp */ \| | |
| | | P94 /* VTp N VT */ \| | |
| | | P137 /* ? */ | = /* ? */ |
| C32 | := | WALK \| LIVE \| | |
| | | P1 /* VTp Np RC */ \| | |
| | | P61 /* VTp Np */ \| | |
| | | P88 /* VTp Ns RC */ \| | |
| | | P122 /* VTp Ns*/ | = /* VPp */ |

| | | | |
|---|---|---|---|
| C52 | := | P41 /* Ns R */ | |
| C56 | := | P36 /* Np R */ | |
| C58 | := | P28 /* Ns VTs */ \| | |
| | | P34 /* Np VTp */ \| | |
| | | P68 /* Ns RC VTs */ \| | |
| | | P147 /* Np RC VTp */ | = /* N VT */ |
| C69 | := | P206 /* Ns R VPs */ \| | = /* Ns RCs */ |
| | | P238 /* Ns R N VT */ | |
| C74 | := | P219 /* Np R VPp */ \| | = /* Np RCp */ |
| | | P249 /* Np R N VT */ | |
| C77 | := | P141 /* Np VPp */ \| | |
| | | P217 /* Np RC VPp */ | = /* NPp VPp */ |
| C119 | := | P148 | = /* VTs N VT */ |
| C122 | := | P243 | = /* Ns R VTs N VT */ |
| C139 | := | P10 /* VTs NPs VPs */ \| | = /* VPs' VPp/s ?*/ |
| | | P32 /* VTs NPp VPp */ | |

where
RCs  =  R VPs | R N VT
RCp  =  R VPp | R N VT
NPp  =  Np | Np RCp
NPs  =  Ns | Ns RCs

## Table 2. Acquired production rules

| | | | |
|---|---|---|---|
| P0 | := C20 /* VTs */ | C74 /* Np RCp */ | = /* VTs Np RCp */ |
| P1 | := C16 /* VTp */ | C74 /* Np RCp */ | = /* VTp Np RCp */ |
| P2 | := C16 /* VTp */ | C77 /* NPp VPp */ | = /* VTp NPp VPp */ |
| P10 | := C20 /* VTs */ | C29 /* NPs VPs */ | = /* VTs NPs VPs */ |
| P28 | := C13 /* Ns */ | C20 /* VTs */ | = /* Ns VTs */ |
| P32 | := C20 /* VTs */ | C77 /* NPp VPp */ | = /* VTs NPp VPp */ |
| P34 | := C26 /* Np */ | C16 /* VTp */ | = /* Np VTp */ |
| P36 | := C26 /* Np */ | C18 /* R */ | = /* Np R */ |
| P41 | := C13 /* Ns */ | C18 /* R */ | = /* Ns R */ |
| P61 | := C16 /* VTp */ | C26 /* Np */ | = /* VTp Np */ |
| P68 | := C69 /* Ns RCs */ | C20 /* VTs */ | = /* Ns RCs VTs */ |
| P74 | := C20 /* VTs */ | C69 /* Ns RCs */ | = /* VTs Ns RCs */ |
| P88 | := C16 /* VTp */ | C69 /* Ns RCs */ | = /* VTp Ns RCs */ |
| P93 | := C69 /* Ns RCs */ | C4 /* VPs */ | = /* Ns RCs VPs */ |
| P94 | := C16 /* VTp */ | C58 /* N VT */ | = /* VTp N VT */ |
| P121 | := C20 /* VTs */ | C13 /* Ns */ | = /* VTs Ns */ |
| P122 | := C16 /* VTp */ | C13 /* Ns */ | = /* VTp Ns */ |
| P137 | := C122 /* Ns R VTs N VT */ | C32 /* VPp */ | = /* ? */ |
| P138 | := C13 /* Ns */ | C4 /* VPs */ | = /* Ns VPs / |
| P141 | := C26 /* Np */ | C32 /* VPp */ | = /* Np VPp */ |
| P147 | := C74 /* Np RCs */ | C16 /* VTp */ | = /* Np RCs VTp */ |
| P148 | := C20 /* VTs */ | C58 /* N VT */ | = /* VTs N VT */ |
| P157 | := C20 /* VTs */ | C26 /* Np */ | = /* VTs Np */ |
| P206 | := C52 /* Ns R */ | C4 /* VPs */ | = /* Ns R VPs */ |
| P217 | := C74 /* Np RCs */ | C32 /* VPp */ | = /* Np RCs VPp */ |
| P219 | := C56 /* Np R */ | C32 /* VPp */ | = /* Np R VPp */ |
| P238 | := C52 /* Ns R */ | C58 /* N VT */ | = /* Ns R N VT */ |
| P243 | := C52 /* Ns R */ | C119 /* VTs N VT */ | = /* (Ns R VTs N) VT */ |
| P249 | := C56 /* Np R */ | C58 /* N VT */ | = /* Np R N VT */ |

## Acknowledgements

The author wishes to thank Prof. Dan Bullock, Prof. Cathy Harris, Prof. Mike Cohen, and Chris Myers of Boston University for valuable discussions.

This work was supported in part by the Air Force Office of Scientific Research (AFOSR F49620-92-J-0225).

## References

[Elman, 1991] Elman, J. (1991). Distributed representations, simple recurrent networks, and grammatical structure. *Machine Learning, 7.*

[Grossberg, 1978] Grossberg, S. (1978). A theory of human memory: Self-organization and performance of sensory-motor codes, maps, and plans. *Progress in Theoretical Biology, 5.*

[John and McClelland, 1988] John, M. F. S. and McClelland, J. L. (1988). Applying contextual constraints in sentence comprehension. In Touretzky, D. S., Hinton, G. E., and Sejnowsky, T. J., editors, *Proceedings of the Second Connectionist Models Summer School 1988, Los Altos, CA.* Morgan Kaufmann Publisher, Inc.

[Mannes, 1993] Mannes, C. (1993). Self-organizing grammar induction using a neural network model. In Mitra, J., Cabestany, J., and Prieto, A., editors, *New Trends in Neural Computation : Lecture Notes in Computer Science 686.* Springer Verlag, New York.

[Miikkulainen and Dyer, 1988] Miikkulainen, R. and Dyer, M. G. (1988). Encoding input/output representations in connectionist cognitive systems. In Touretzky, D. D., Hinton, G. E., and Senowsky, T. J., editors, *Proceedings of the Second Connectionist Models Summer School 1988, Los Altos, CA.* Morgan Kauffman Publisher, Inc.

[Ritter and Kohonen, 1990] Ritter, H. and Kohonen, T. (1990). Learning semanto-topic maps from context. *Proceedings of . IJCNN 90, Washington D.C., I.*

[Scholtes, 1991] Scholtes, J. C. (1991). Unsupervised context learning in natural language processing. *Proceedings of IJCNN Seattle 1991.*

## Appendix A. Activation and learning equations

### A.1 Classification Network Activities

•*Gradient Field*
$$X0_i(t) = 0.5X0_i(t-1) + I_i(t) \tag{1}$$

where $t$ is a discrete time, $i$ is the symbol id. and $I_i(t)$ is an input symbol.

•*Input Layer*
$$X1_{Ai}(t) = \theta(2(X0_i(t)-\theta(X0_i(t)))), \quad X1_{Bi}(t) = \theta(X0_i(t)), \quad X1_{Ci}(t) = I_i(t+1)$$

Where the suffix $A$, $B$, and $C$ the most recent, the next to most recent, and the lookahead symbols, respectively. Weights in networks A, B, and C are identical.

$$\theta(x) = \begin{cases} 1 & \text{if } x > 1 - 2^{-M} \\ 0 & \text{otherwise} \end{cases}$$

Here $M$ is the maximum number of symbols on the gradient field.

•*Feature Layer*

$$X2^I_{si} = \sum_j X1_{sj} W1_{sji}, \quad X2^{II}_{si}{}' = f(X1^I_{si}/(a+\sum_j X2^I_{sj})), \quad X2_{si} = X2^{II}_{si}/(a+\sum_j X2^{II}_{sj})$$

$$f(x) = 2/(1+exp(-Tx)) - 1$$

where $s$ is a suffix which is either $A$, $B$, or $C$ and $T$ is the steepness of the sigmoid function and $a$ is a small positive constant. Table 4 shows the meaning of above suffix $i$.

•*Category Layer*

$$X3_{pi} = \begin{cases} 1 & \text{if } i = min\{j|\sum_{ks} X2_{sk} W2_{skj} > \rho\}, \text{ or} \\ & \text{if } \phi = min\{j|\sum_{ks} X2_{sk} W2_{skj} > \rho\} \text{ \& } unref_i =^{max}_j \{unref_j\} \\ 0 & \text{otherwise} \end{cases} \quad (2)$$

Where $\rho$ is the least match score required and $uref_i$ is an unreferenced count.

## A.2  Classification Learning

•*Feature Weights*
$$\Delta W1_{sij} = -\alpha_1 W1_{sij} + \beta_1 X1_i(X2_{sj} - W1_{sij})$$
where $\alpha_1$ is the forgetting rate, and $\beta_1$ is the learning rate.

•*Categorization Weights*

$$\begin{cases} \Delta W2_{sij} = \beta_2 X3_{si}(X2_{si} - W2_{sij}) & \text{if the node is selected by the first line of (2)} \\ W2_{sij} = X2_{si} & \text{if the node is selected by the second line of (2)} \end{cases}$$

where $\beta_2$ is the learning rate.

## A.3  Production Network Activities

•*Mutual predictiveness*

$$
\begin{array}{llllll}
X4_{ij} & = X3_{Ai} W3_{ij}, & X5_{ji} & = X3_{Bj} W4_{ji}, & X6_{ij} & = X4_{ij} X5_{ji} \\
X7_{ij} & = X3_{Bi} W3_{ij}, & X8_{ji} & = X3_{Cj} W4_{ji}, & X9_{ij} & = X7_{ij} X8_{ji}
\end{array}
$$

The phrase identification number for a category pair $(i,j)$ is given algorithmically in the current version by a cash function $cash(i,j)$.

(i) Case in which $\gamma \sum_{ij} X6_{ij} \geq \sum_{ij} X9_{ij}$  :  Reduce

$$X10_i = \begin{cases} 1 & \text{if } i = cash(I,J) \text{ where } X6_{IJ} =^{max}_{ij} (X6_{ij}) \\ 0 & \text{otherwise} \end{cases}$$

$$X0_i(t+1) = 0.5 * pop(pop(X0_i(t))) + X10, \quad pop(x) = 2(x - \theta(x))$$

(ii) Case in which $\gamma \sum_{ij} X6_{ij} < \sum_{ij} X9_{ij}$  :  Shift

The next input symbol is added on the gradient field, as was expressed in (1).

**A.4 Production Learning**

$$\Delta W3_{ij} = -\alpha_3 W3_{ij} + \beta_3 X3_{Ai}(X3_{Bj} - W3_{ij}), \quad \Delta W4_{ji} = -\alpha_3 W4_{ji} + \beta_3 X3_{Bj}(X3_{Ai} - W4_{ji})$$

where $X3_{Ai}$ and $X3_{Bj}$ are nodes that receive the next to the most recent symbol $i$ and the most recent symbol $j$, respectively.

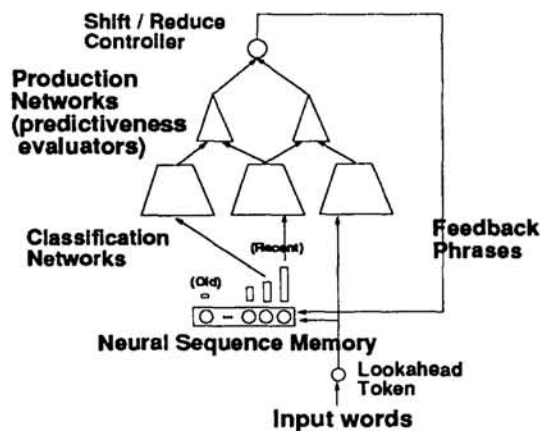

Figure 1. Block diagram of the network

| S | → | NP VP . |
|---|---|---|
| NP | → | N \| N RC |
| VP | → | V [NP] |
| RC | → | who NP V \| who VP |
| N | → | boy \| girl \| cat \| dog \| |
| | | boys \| girls \| cats \| dogs |
| V | → | chase \| feed \| work \| live \| |
| | | chases \| feeds \| works \| lives |

Number agreement
- Agreements between N and V within clause
- Agreements between head N and subordinate V (where appropriate)

Verb arguments
- chase, feed -> require a direct object
- walk, live -> preclude a direct object
(Observed also for head/verb relations in relative clauses)

Table 3. Grammar for generated sentences

| (1) | Left context, words. $s = L, 1 <= i <= N_w$ |
|---|---|
| (2) | Left context, phrases. $s = L, N_w < i <= N_w + N_p$ |
| (3) | Left context, categories. $s = L, N_w + N_p < i <= N_w + N_p + N_c$ |
| (4) | Right context, words. $s = R, 1 <= i <= N_w$ |
| (5) | Right context, phrases. $s = R, N_w < i <= N_w + N_p$ |
| (6) | Right context, categories. $s = R, N_w + N_p < i <= N_w + N_p + N_c$ |
| (7) | Right context, lookahead. $s = R, N_w + N_p + N_c < i <= 2N_w + N_p + N_c$ |

$N_w$, $N_p$, and $N_c$ denotes number of words, phrases, and categories, respectively.

Table 4. Subfields in a feature layer

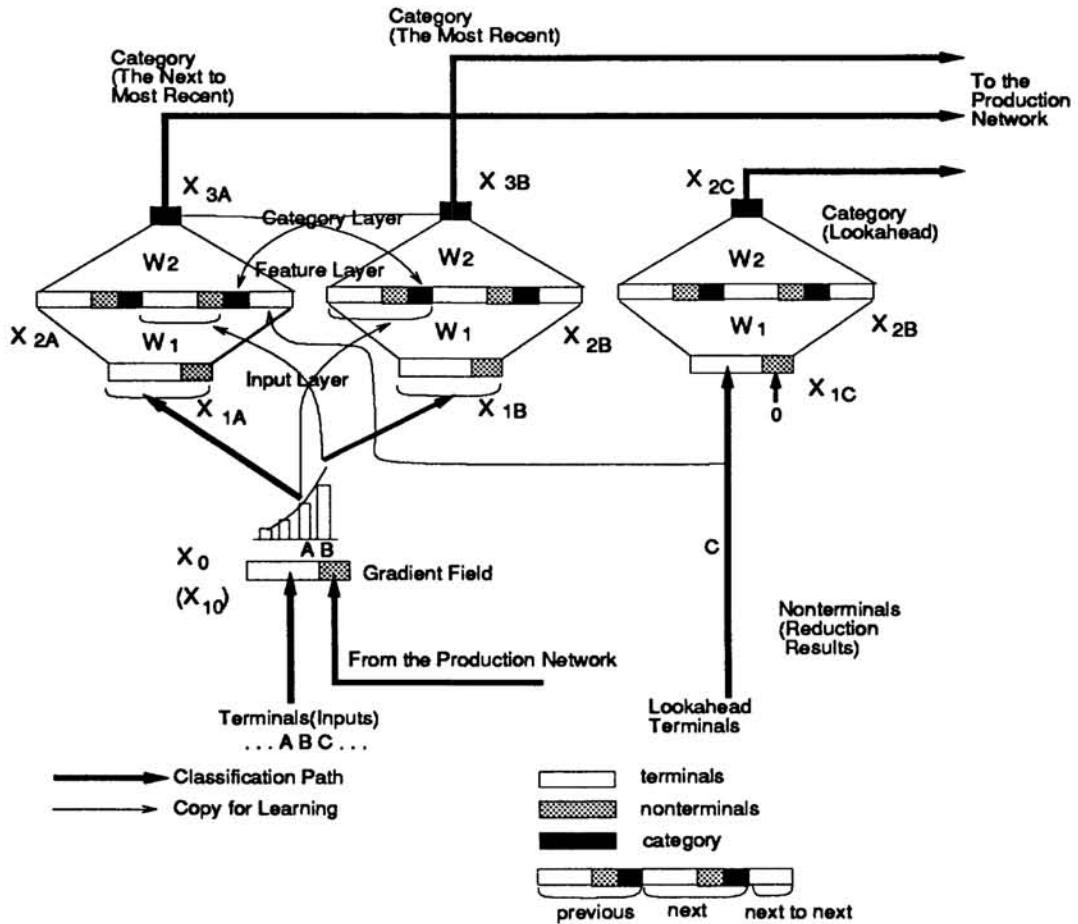

Figure 2. Classification Network

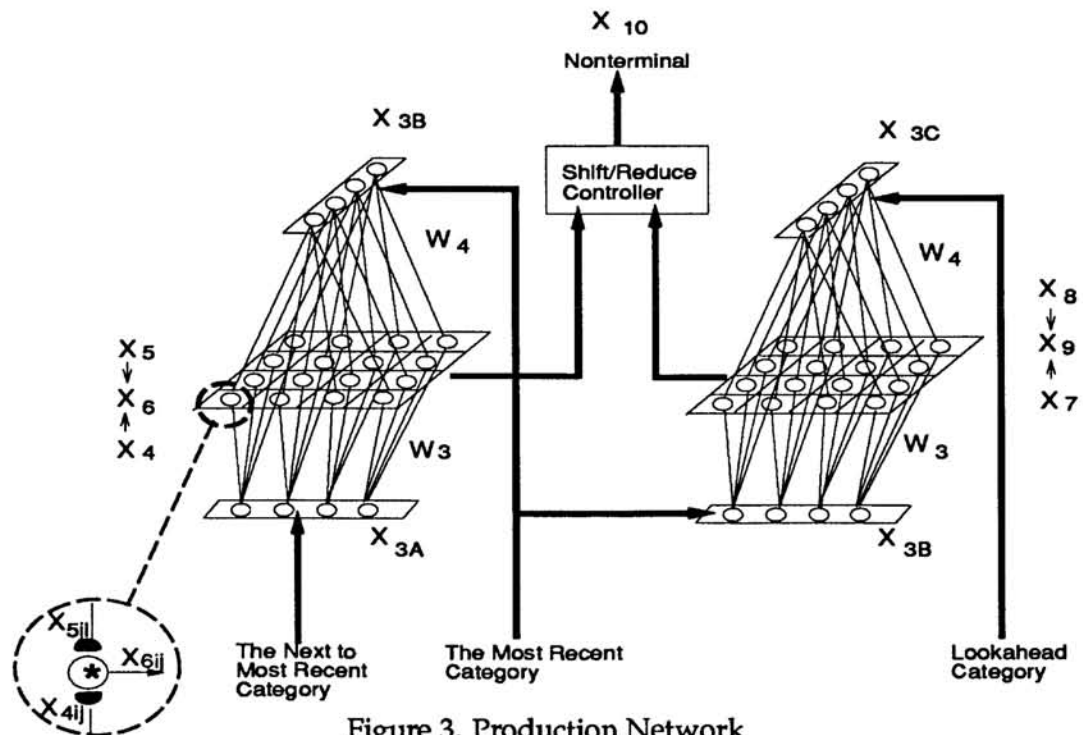

Figure 3. Production Network